# A Complexity-Distortion Approach to Joint Pattern Alignment

**Andrea Vedaldi    Stefano Soatto**
Department of Computer Science
University of California at Los Angeles
Los Angeles, CA 90035
{vedaldi,soatto}@cs.ucla.edu

## Abstract

Image Congealing (IC) is a non-parametric method for the joint alignment of a collection of images affected by systematic and unwanted deformations. The method attempts to undo the deformations by minimizing a measure of complexity of the image ensemble, such as the averaged per-pixel entropy. This enables alignment without an explicit model of the aligned dataset as required by other methods (e.g. transformed component analysis). While IC is simple and general, it may introduce degenerate solutions when the transformations allow minimizing the complexity of the data by collapsing them to a constant. Such solutions need to be explicitly removed by regularization.

In this paper we propose an alternative formulation which solves this regularization issue on a more principled ground. We make the simple observation that alignment should simplify the data *while preserving the useful information* carried by them. Therefore we trade off fidelity and complexity of the aligned ensemble rather than minimizing the complexity alone. This eliminates the need for an explicit regularization of the transformations, and has a number of other useful properties such as noise suppression. We show the modeling and computational benefits of the approach to the some of the problems on which IC has been demonstrated.

## 1   Introduction

Joint pattern alignment attempts to remove from an ensemble of patterns the effect of nuisance transformations of a systematic nature. The aligned patterns have then a simpler structure and can be processed more easily. Joint pattern alignment is not the same problem as aligning a pattern to another; instead all the patterns are projected to a common "reference" (usually a subspace) which is unknown and needs to be discovered in the process.

Joint pattern alignment is useful in many applications and has been addressed by several authors. Here we only review the methods that are most related the present work.

Transform Component Analysis [7] (TCA) explicitly models the aligned ensemble as a Gaussian linear subspace of patterns. In fact, TCA is a direct extension of Probabilistic Principal Component Analysis (PPCA) [10]: Patterns are generated as in standard PPCA and additional hidden layers model the nuisance deformations. Expectation-maximization is used to learn the model from data which result in their alignment. Unfortunately the method requires the space of transformations to be quantized and it is not clear how well the approach could scale to complex scenarios.

Image Congealing (IC) [9] takes a different perspective. The idea is that, as the nuisance deformations should increase the complexity of the data, one should be able to identify and undo them by

contrasting this effect. Thus IC transforms the data to minimize an appropriate measure of the "complexity" of the ensemble. With respect to TCA, IC results in a lighter formulation which enables addressing more complex transformations and makes fewer assumptions on the aligned ensemble.

An issue with the standard formulation of IC is that it does not require the aligned data to be a faithful representation of the original data. Thus simplifying the data might not only remove the nuisance factors, but also the useful information carried by the patterns. For example, if entropy is used to measure complexity, a typical degenerate solution is obtained by mapping all the data to a constant, which results in minimum (null) entropy. Such solutions are avoided by explicitly regularizing the transformations, in ways that are however rather arbitrary [9].

One should instead search for an optimal compromise between complexity of the simplified data and preservation of the useful information (Sect. 2). This approach is not only more direct, but also conceptually more straightforward as no *ad hoc* regularization needs to be introduced. We illustrate some of its relationship with rate-distortion theory (Sect. 2.1) and information bottleneck [2] (Sect. 2.2) and we contrast it to IC (Sect. 2.4).

In Sect. 3 we specialize our model to the problem of image alignment as done in [9]. For this case, we show that the new model has the same computational complexity of IC (Sect. 3.1). We also show that a Gauss-Newton based algorithm is possible, which is useful to converge quickly during the final stage of the optimization (Sect. 3.2; in a similar context a descent based algorithm was introduced in [1]). In Sect. 4 we illustrate the practical behavior of the algorithm, showing how the complexity-distortion compromise affects the final solution. In particular, our results compare favorably with the ones of [9], with the added simplicity and other benefits, such as noise suppression.

## 2  Problem formulation

We formulate joint pattern alignment as the problem of finding a deformed pattern ensemble which is simpler but faithful to the original data. This is similar to a lossy compression problem [5, 4, 3] and is in fact equivalent to it in some cases (Sect. 2.1).

A *pattern (or data) ensemble* $x \in \mathcal{X}$ is a random variable with density $p(x)$. Similarly, an *aligned ensemble* or *alignment* $y \in \mathcal{X}$ of the ensemble $x$ is another variable $y$ that has conditional statistic $p(y|x)$. We seek for an alignment that is "simpler" than $x$ but "faithful" to $x$. The complexity $R$ of the alignment $y$ is measured by an operator $R = H(y)$ such as, for example, the entropy of the random variable $y$ (but we will see other options). The cost of representing $x$ by $y$ is expressed by a *distortion function* $d(x, y) \in \mathbb{R}_+$ and the faithfulness of the alignment $y$ is quantified as the expected *distortion* $D = E[d(x, y)]$.

Consider a class $\mathcal{W}$ of deformations $w : \mathcal{X} \to \mathcal{X}$ acting on the patterns $\mathcal{X}$. In order for the alignment $y$ to factor out $\mathcal{W}$ we consider a distortion function which is invariant to the action of $\mathcal{W}$; in particular, given a base distortion $d_0(x, y)$, we consider the *deformation invariant distortion*

$$d(x, y) = \min_{w \in \mathcal{W}} d_0(x, w(y))$$

Thus an aligned pattern $y$ is faithful to a deformed pattern $x$ if it is possible to map $y$ to $x$ by a nuisance deformation $w$.

Figuring out the best alignment $y$ boils down in optimizing $p(y|x)$ for complexity and distortion. However, this require trading off complexity and distortion and there is no unique way of doing so. The *distortion-complexity function* $D(R)$ gives the best distortion $D$ that can be achieved by alignments of complexity $R$. All such distortion-optimal alignments are equally good in principle, and it is the application that poses an upper bound on the acceptable distortion.

$D(R)$ can be computed by optimizing the distortion $D$ w.r.t. $p(y|x)$ while keeping constant the complexity $R$. However it is usually easier optimize the Lagrangian

$$\min_{p(y|x)} D + \lambda R \tag{1}$$

whose optimum is attained where the derivative of $D(R)$ is equal to $-\lambda$. Then by varying $\lambda$ one spans the graph of $D(R)$ and finds all the optimal alignments for given complexities.

## 2.1 Relation to rate-distortion and entropy constrained vector quantization

If one chooses the mutual information $I(x, y)$ as complexity measure $H(y)$ in eq. (1), then (1) becomes a *rate-distortion* problem and the function $D(R)$ a *rate-distortion function* [5]. The formulation is valid both for discrete and continuous spaces $\mathcal{X}$, but yields to a mapping $p(y|x)$ that is genuinely stochastic. Therefore the alignment $y$ of a pattern $x$ is in general not unique. This is because in rate-distortion $y$ is an auxiliary variable used to derive a deterministic code for *long sequences* $(x_1, \ldots, x_n)$ of data, not for data $x$ in isolation.

In contrast, *entropy constrained vector quantization* [4, 3] assumes that $y$ is finite (i.e. that it spans a finite subset of $\mathcal{X}$) and that it is functionally determined by $x$ (i.e. $y = y(x)$). Then it measures the complexity of $y$ as the (discrete) entropy $H(y)$. This is analogous to a rate-distortion problem, except that one searches for a "single letter" optimal coding $y$ of $x$ rather than an optimal coding for long sequences $(x_1, \ldots, x_n)$. Unlike rate-distortion, however, the aligned ensemble $y$ is discrete even if the ensemble $x$ is continuous.

## 2.2 Relation to information bottleneck

Information Bottleneck (IB) [2] is a special rate-distortion problem in which one compresses a variable $x$ while preserving the information carried by $x$ about another variable $z$, representing the task of interest. In this sense IB is similar to the idea proposed here. By designing an appropriate distribution $p(x, z)$ it may also be possible to obtain an alignment effect similar to the one we seek here. For example, if $\mathcal{W}$ is a group of transformations, one may define $z = z(x) = \{w(x) : w \in \mathcal{W}\}$, for which $z$ is indifferent exactly to the deformations $w$ of $x$.

## 2.3 Alternative measures of complexity

Instead of the entropy $H(y)$ or the mutual information $I(x, y)$ we can use alternative measures of complexity that yield to more convenient computations. An example is the averaged-per-pixel entropy introduced by IC [9] and discussed in Sect. 3. Generalizing this idea, we assume that the aligned data $y$ depend functionally on the patterns $x$ (i.e. $y = y(x)$) and we express the complexity of $y$ as the total entropy of lower dimensional projections $\phi_1(y), \ldots, \phi_M(y)$, $\phi_i : \mathcal{X} \to \mathbb{R}^k$ of the ensemble.

Distortion and entropies are estimated empirically and non-parametrically. Concretely, given an ensemble $x_1, \ldots, x_K \in \mathcal{X}$ of patterns, we recover transformations $w_1, \ldots, w_K \in \mathcal{W}$ and aligned patterns $y_1, \ldots, y_K \in \mathcal{X}$ that minimize

$$\frac{1}{K} \sum_{i=1}^{K} d(x_i, w_i(y_i)) - \lambda \sum_{j=1}^{M} \frac{1}{K} \sum_{i=1}^{K} \log p_j(\phi_j(y_i)),$$

where the densities $p_j(\phi_j(y))$ are estimated from the samples $\phi_j(y_1), \ldots, \phi_j(y_K)$ by histogramming (discrete case) or by a Parzen estimator [6] with Gaussian kernel $g_\sigma(y)$ of variance $\sigma$ (continuous case[1]), i.e.

$$p_j(\phi_j(y)) = \frac{1}{N} \sum_{i=1}^{N} g_\sigma(\phi_j(y) - \phi_j(y_i)).$$

## 2.4 Comparison to image congealing

In IC [9], given data $x_1, \ldots, x_K \in \mathcal{X}$, one looks for transformations $v : \mathcal{X} \to \mathcal{X}$, $x \mapsto y$ such that the density $p(y)$ estimated from samples $y_1 = v_1(x_1), \ldots, y_K = v_K(x_K)$ has minimum entropy. If the transformations enable to do so, one can minimize the entropy by mapping all the patterns to a constant; to avoid this one considers the regularized cost function

$$H(y) + \alpha \sum_i R(v_i) \tag{2}$$

where $R(v)$ is a term penalizing unacceptable deformations. Compared to IC, in our formulation:

▶ The distortion term $E[d(x, y)]$ substitutes the arbitrary regularization $R(v)$.

▶ The aligned patterns $y$ are not obtained by deforming the patterns $x$; instead $y$ is obtained as a *simplification* of $x$ within an acceptable level of distortion. This fact induces a noise-cancellation effect (Sect. 4).

▶ The transformations $w$ can be rather general, even non-invertible. IC can use complex transformations too, but most likely these would need to be heavily regularized as they would tend to annihilate the patterns.

## 3  Application to joint image alignment

We apply our model to the problem of removing a family of geometric distortions from images. This is the same application for which IC [7] was proposed in the first place.

We are given a set $I_1(x), \dots, I_K(x)$ of digital images (pattern ensemble) defined on a regular lattice $x \in \Lambda \subset \mathbb{R}^2$ and with range in $[0, 1]$. The images may be affected by parametric transformations $w_i(\cdot) = w(\cdot; q_i) : \mathbb{R}^2 \to \mathbb{R}^2$, so that

$$I_i(x) = T_i(wx) + n_i(x), \quad x \in \Lambda$$

for templates (aligned ensemble[2]) $T_i(y)$, $y \in \Lambda$ and residuals $n_i(x)$. Here $q_i$ is the vector of parameters of the transformation $w_i$ (for example, $w_i$ might be a 2-D affine transformation $y = Lx + l$ and $q_i$ the vector $q = [L_{11} \quad L_{21} \quad L_{12} \quad L_{22} \quad l_1 \quad l_2]$).

The templates $T_i(y)$, $y \in \Lambda$ are digital images themselves. In order to define $T_i(wx)$ when $wx \notin \Lambda$, bilinear interpolation and zero-padding are used. Therefore the symbol $T_i(w_i x)$ really denotes the quantity

$$T(w_i x) = A(x; w_i) T_i, \qquad x \in \Lambda$$

where $A(x; w_i)$ is a row vector of mixing coefficients determined by $w_i$ and and the interpolation method being used and $T_i$ is the vector obtained by stacking the pixels of the template $T_i(y)$, $y \in \Lambda$. We will also use the notation $w_i \circ T_i = A(w_i) T_i$ where the left hand side is the stacking of the warped template $T(w_i x)$, $x \in \Lambda$ and $A(w_i)$ is the matrix whose rows are the vectors $A(x; w_i)$ for $x \in \Lambda$.

The distortion is defined to be the squared $l^2$ norm of the residual $d(I_i, w \circ T_i) = \sum_{x \in \Lambda} (I_i(x) - T_i(w_i x))^2$. The complexity of the aligned ensemble $T(y)$, $y \in \Lambda$ is computed as in Sect. 2.3 by projecting on the image pixels and averaging their entropies (this is equivalent to assuming that the pixels are statistically independent). For each pixel $y \in \Lambda$ a density $p(T(y) = t)$, $t \in [0, 1]$ is estimated non parametrically from the data $\{T_1(y), \dots, T_K(y)\}$ (we use Parzen window as explained in Sect. 2.3). The complexity of a pixel is thus

$$H(T(y)) = -\frac{1}{K} \sum_{i=1}^{K} \log p(T_i(y)).$$

Finally the overall cost function is obtained by summing over all pixels and averaging over all images:

$$L(w_1, \dots, w_K, T_1, \dots, T_K) = \frac{1}{K} \sum_{i=1}^{K} \sum_{x \in \Lambda} (I_i(x) - T_i(w_i x))^2 - \lambda \frac{1}{K} \sum_{i=1}^{K} \sum_{y \in \Lambda} \log p(T_i(y)). \quad (3)$$

### 3.1  Basic search

In this section we show how the optimization algorithm from [7] can be adapted to work with the new formulation. This algorithm is a simple coordinate maximization in the dimensions of the search space:

1: Estimate the probabilities $p(T(y))$, $y \in \Lambda$ from the templates $\{T_i(y) : i = 1, \ldots, K\}$
2: For each pattern $i = 1, \ldots, K$ and for each component $q_{ji}$ of the parameter vector $q_i$, try a few values of $q_{ji}$. For each value re-compute the cost function (3) and keep the best.
3: Repeat, refining the sampling step of the parameters.

This algorithm is appropriate if the dimensionality of the parameter vector $q$ is reasonably small. Here we consider affine transformations for the sake of the illustration, so that $q$ is six-dimensional.

In (1.) and (2.) estimating the probabilities $p(T_i(y))$ and the cost function $L(w_1, \ldots, w_K, T_1, \ldots, T_K)$ requires to know $T_i(y)$. As a first order approximation (as the final result will be refined by Gauss-Newton as explained in the next Section), we bypass this problem and we simply set $T_i = w_i^{-1} \circ I_i$, exploiting the fact that the affine transformations $w_i$ are invertible[3]. Eventually all we do is substituting the regularization term $\sum_i R(v_i)$ of [9] with the expected distortion

$$\frac{1}{K} \sum_{i=1}^{K} \sum_{x \in \Lambda} (I_i(x) - w_i \circ (w_i^{-1} \circ I_i(x)))^2 = \frac{1}{K} \sum_{i=1}^{K} \sum_{x \in \Lambda} (I_i(x) - A(x; w_i) A(w_i^{-1}) I_i)^2$$

Note that warping and un-warping the image $I_i$ is a lossy operation even if $w_i$ is bijective because the transformation, applied to digital images, introduces aliasing. Thus the new algorithm is simply avoiding those transformations $w_i$ that would introduce excessive loss of fidelity.

## 3.2 Gauss-Newton search

With respect to IC, where only the transformations $w_1, \ldots, w_K$ are estimated, here we compute the templates $T_1, \ldots, T_k$ as well. While this might be not so important when a coarse approximation to the solution has to be found (for which the algorithm of Sect. 3.1 can be used), it must be taken into account to get refined results. This can be done (with a bit of numeric care) by Gauss-Newton (GN).

Applying Gauss-Newton requires to take derivatives with respect to the pixel values $T_i(y)$. We exploit the fact that the variables $T(y)$ are continuous, as opposed to [9].

We still process a single image per time, reiterating several times across the whole ensemble $\{I_1(x), \ldots, I_K(x)\}$. For a given image $I_i$ we update the warp parameters $q_i$ and the template $T_i$ simultaneously. We exploit the fact that, as the number $K$ of images is usually big, the density $p(T(y))$ does not change significantly when only one of the templates $T_i$ is being changed. Therefore $p(T(y))$ can be assumed constant in the computation of the gradient and the Hessian of the cost function (3). The gradient is given by

$$\frac{\partial L}{\partial q_i^\top} = \sum_{x \in \Lambda} 2 \Delta_i(x) \nabla T_i(w_i x) \frac{\partial w_i}{\partial q_i^\top}(x), \qquad \frac{\partial L}{\partial T_i(y)} = \sum_{x \in \Lambda} 2 \Delta_i(x)(A(x; w_i)\delta_y) - \sum_{y \in \Lambda} \frac{\dot{p}(T_i(y))}{p(T_i(y))}$$

where $\Delta_i(x) = T_i(w_i x) - I_i(x)$ is the reconstruction residual, $A(x; w_i)$ is the linear map introduced in Sect. 3 and $\delta_y = \delta(z - y)$ is the 2-D discrete delta function centered on $y$, encoded as a vector.

The approximated Hessian of the cost function (3) can be obtained as follows. First, we use the Gauss-Newton approximation for the derivative w.r.t. the transformation parameters $q_i$

$$\frac{\partial^2 L}{\partial q_i \partial q_i^\top} \approx \sum_{x \in \Lambda} 2 \frac{\partial w_i^\top}{\partial q_i}(x) \nabla^\top T_i(w_i x) \nabla T_i(w_i x) \frac{\partial w_i}{\partial q_i^\top}(x)$$

We then have

$$\frac{\partial^2 L}{\partial T_i(y)^2} = \sum_{x \in \Lambda} 2(A(x; w_i)\delta_y)^2 - \sum_{y \in \Lambda} \frac{\ddot{p}(T_i(y))p(T_i(y)) - \dot{p}(T_i(y))^2}{p(T_i(y))^2}$$

$$\frac{\partial^2 L}{\partial T_i(y) \partial T_i(z)} = \sum_{x \in \Lambda} 2(A(x; w_i)\delta_y)(A(x; w_i)\delta_z)$$

$$\frac{\partial^2 L}{\partial T_i(y) \partial q^\top} = \sum_{x \in \Lambda} 2(A(x; w_i)\delta_y) \nabla T_i(w_i x) \frac{\partial w_i}{\partial q_i^\top} + \sum_{x \in \Lambda} 2\Delta_i(x) A(x; w_i) \begin{bmatrix} D_1 \delta_y & D_2 \delta_y \end{bmatrix} \frac{\partial w_i}{\partial q_i^\top}$$

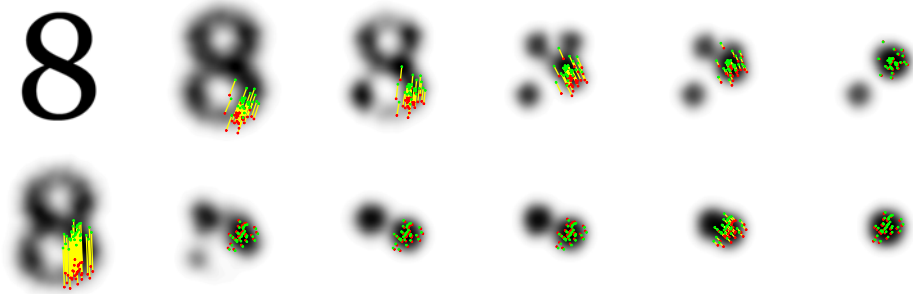

Figure 1: *Toy example.* **Top left.** We distort the patterns by applying translations drawn uniformly from the 8-shaped region (the center corresponds to the null translation). **Top.** We show the gradient based algorithm while it gradually aligns the patterns by reducing the complexity of the alignment $y$. Dark areas correspond to high values of the density of the alignment; we also superimpose the trajectory of one of the patterns. Unfortunately the gradient based algorithm, being a local technique, gets trapped in two local modes (the modes can however be fused in a post-processing stage). **Bottom.** The basic algorithm completely eliminates the effect of the nuisance transformations doing a better job of avoiding local minima. Although for this simple problem the basic search is more effective, on more difficult scenarios the extra complexity of the Gauss-Newton search pays off (see Sect. 4).

where $D_1$ is the discrete linear operator used to compute the derivative of $T_i(y)$ along its first dimension and $D_2$ the analogous operator for the second dimension. The second term of the last equation gives a very small contribution and can be dropped.

The equations are all straightforward and result in la linear system

$$\delta\theta^\top \left( \frac{\partial^2 L}{\partial\theta\partial\theta^\top} \right) = -\frac{\partial L}{\partial\theta^\top}$$

where the vector $\theta^\top = \left[ q^\top T(y_1)\ldots T(y_n) \right]$ has size in the order of the number of pixels of the template $T(y)$, $y \in \Lambda$. While this system is large, it is also extremely sparse an can be solved rather efficiently by standard methods [8].

## 4    Experiments

The first experiment (Fig.1) is a toy problem illustrating our method. We collect $K$ patterns $x_i$, $i = 1,\ldots,K$ which are arrays of $M$ 2D points $x_i = (x_{1i},\ldots,x_{Mi})$. Such points are generated by drawing $M$ i.i.d. samples from a 2-D Gaussian distribution and adding a random translation $w_i \in \mathbb{R}^2$ to them. The distribution of the translations $w_i$ is generic (in the example $w_i$ is drawn uniformly from an 8-shaped region of the plane): This is not a problem as we do not need to make any particular assumptions on $w$ besides that it is a translation. The distortion $d(x_i, y_i)$ is simply the sum of the Euclidean distances $\sum_{j=1}^{m} \|y_{ji} + w_i - x_{ji}\|^2$ between the patterns $x_i$ and the transformed codes $w_i(y_i) = (y_{1i} + w_i,\ldots,y_{mi} + w_i)$. The distribution $p(y_i)$ of the codes is assumed to factorize as $p(y_i) = \prod_{j=1} p(y_{ji})$ where the $p(y_{ji})$ are identical densities estimated by Parzen window from all the available samples $\{y_{ji}, j = 1,\ldots,M, i = 1,\ldots,K\}$.

In the second experiment (Fig. 2) we align hand-written digits extracted from the NIST Special Database 19. The results (Fig. 3) should be compared to the ones from [9]: They are of analogous quality, but they were achieved without regularizing the class of admissible transformations. Despite this, we did not observe any of the aligned patterns to collapse. In Fig. 4 we show the effect of choosing different values of the parameter $\lambda$ in the cost function (3). As $\lambda$ is increased, the alignment complexity is reduced and the fidelity of the alignment is degraded. By an appropriate choice of $\lambda$, the alignment can be regarded as a "restoration" or "canonization" of the pattern which abstracts from details of the specific instance.

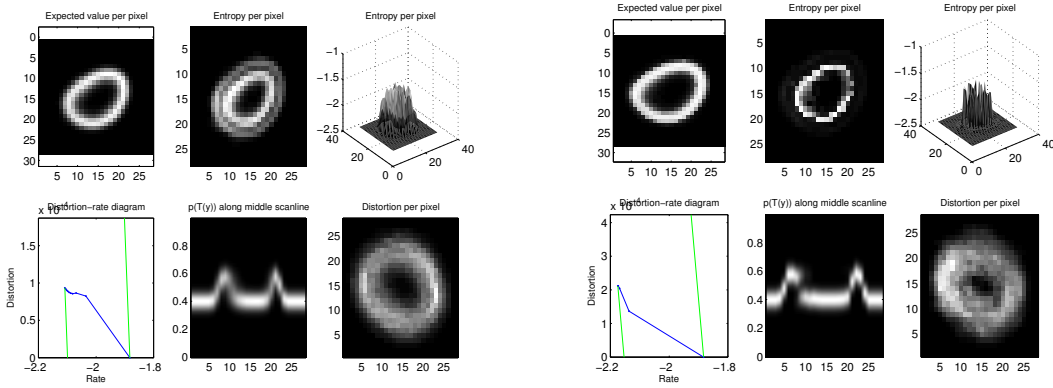

Figure 2: *Basic vs GN image alignment algorithms.* **Left.** We show the results of applying the basic image alignment algorithm of Sect. 3.1. The patterns are zeroes from the NIST Special Database 19. We show in writing order: The expected value $E[T(y)]$; the per-pixel entropy $H(T(y))$ (it can be negative as it is differential); a 3-D plot of the same function $H(T(y))$; the distortion-complexity diagram as the algorithm minimizes the function $D + \lambda R$ (in green we show some lines of constant cost); the probability $p(T(y) = l)$ as $l \in [0, 1]$ and $y$ varies along the middle scan-line; and the per-pixel distortion $D(x) = E[(I(x) - T(wx))^2]$. **Right.** We demonstrate the GN algorithm of Sect. 3.2. The algorithm achieves a significantly better solution in term of the cost function (3). Moreover GN converges in only two sweeps of the dataset, while the basic algorithm after 10 sweeps is still slowly moving. This is due to the fact that GN selects both the best search direction and step size, resulting in a more efficient search strategy.

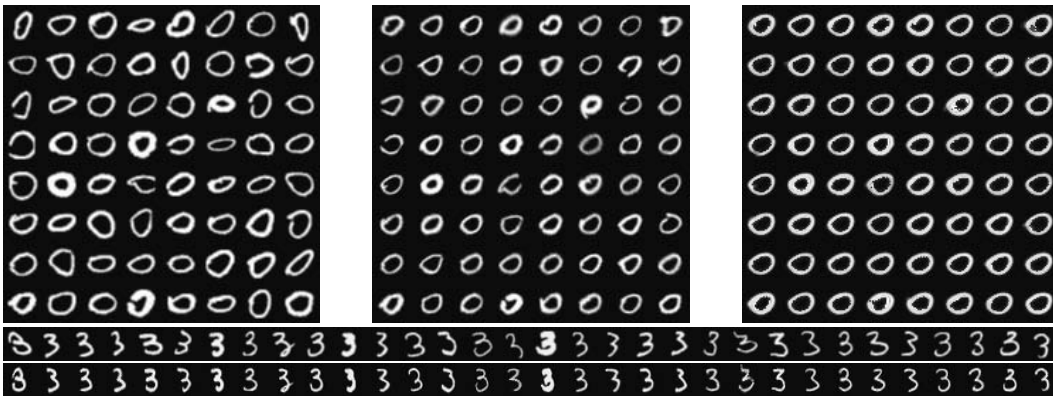

Figure 3: *Aligned patterns.* **Left.** A few patterns from NIST Special Database 19. **Middle.** Basic algorithm: Results are very similar to [9], except that no regularization on the transformations is used. **Right.** GN algorithm: Patterns achieve a better alignment due to the more efficient search strategy; they also appear to be much more "regular" due to the noise cancellation effect discussed in Fig. 4. **Bottom.** More examples of patterns before and after GN alignment.

## 5 Conclusions

IC is a useful algorithm for joint pattern alignment, both robust and flexible. In this paper we showed that the original formulation can be improved by realizing that alignment should result in a *simplified representation of the useful information carried by the patterns* rather than a simplification of the patterns. This results in a formulation that does not require inventing regularization terms in order to prevent degenerate solutions. We also showed that Gauss-Newton can be successfully applied to this problem for the case of image alignment and that this is in some regards more effective than the original IC algorithm.

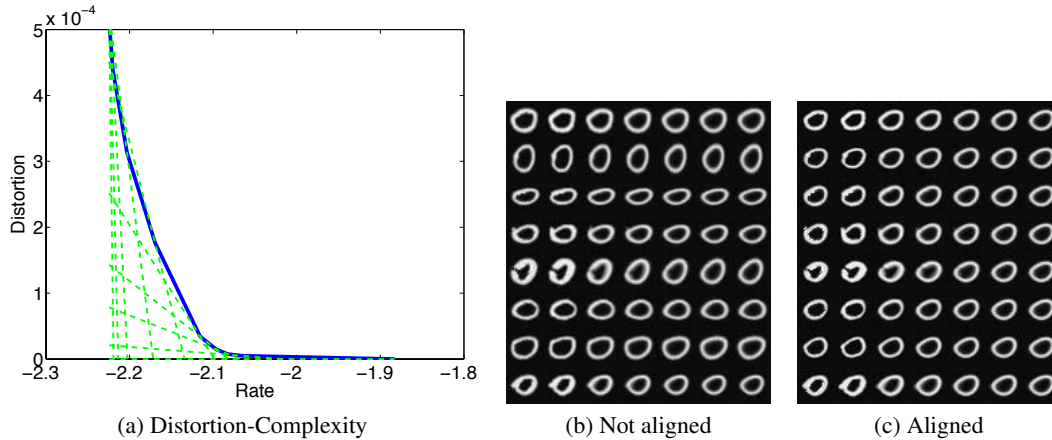

| (a) Distortion-Complexity | (b) Not aligned | (c) Aligned |

Figure 4: *Distortion-complexity balance.* We illustrate the effect of varying the parameter $\lambda$ in (3). **(a)** Estimated distortion-complexity function $D(R)$. The green (dashed) lines have slope equal to $\lambda$ and should be tangent to $D(R)$ (Sect. 2). **(b)** We show the alignment $T(w_i x)$ of eight patterns (rows) as $\lambda$ is increased (columns). In order to reduce the entropy of the alignment, the algorithm "forgets" about specific details of each glyph. **(c)** The same as (b), but aligned.

**Acknowledgments**

We would like to acknowledge the support of AFOSR FA9550-06-1-0138 and ONR N00014-03-1-0850.

## Footnotes

[1]The Parzen estimator implies that the differential entropy of the distributions $p_j$ is always lower bounded by the entropy of the kernel $g_\sigma$. This prevents the differential entropy to have arbitrary small negative values.

[2]With respect to Sect. 2 the patterns $x_i$ are now the images $I_i$ and the alignment $y$ are the templates $T_i$.

[3]Our criterion avoids implicitly non-invertible affine transformations as they yield highly distorted codes.

## References

[1] P. Ahammad, C. L. Harmon, A. Hammonds, S. S. Sastry, and G. M. Rubin. Joint nonparametric alignment for analizing spatial gene expression patterns in drosophila imaginal discs. In *Proc. CVPR*, 2005.

[2] K. Branson. The information bottleneck method. Lecture Slides, 2003.

[3] J. Buhmann and H. Kühnel. Vector quantization with complexity costs. *IEEE Trans. on Information Theory*, 39, 1993.

[4] P. A. Chou, T. Lookabaugh, and R. M. Gray. Entropy-constrained vector quantization. In 37, editor, *IEEE Trans. on Acoustics, Speech, and Signal Processing*, volume 1, 1989.

[5] T. M. Cover and J. A. Thomson. *Elements of Information Theory*. Wiley, 2006.

[6] R. O. Duda, P. E. Hart, and D. G. Stork. *Pattern Classification*. Wiley Inerscience, 2001.

[7] B. J. Frey and N. Jojic. Transformation-invariant clustering and dimensionality reduction using EM. *PAMI*, 2000.

[8] G. H. Golub and C. F. Van Loan. *Matrix Computations*. The Johns Hopkins University Press, 1996.

[9] E. G. Learned-Miller. Data driven image models through continuous joint alignment. *PAMI*, 28(2), 2006.

[10] M. E. Tipping and C. M. Bishop. Probabilistic principal component analysis. *Journal of The Royal Statistical Society, Series B*, 61(3), 1999.
